# Convergence and Consistency of Regularized Boosting Algorithms with Stationary $\beta$-Mixing Observations

**Aurélie C. Lozano**
Department of Electrical Engineering
Princeton University
Princeton, NJ 08544
alozano@princeton.edu

**Sanjeev R. Kulkarni**
Department of Electrical Engineering
Princeton University
Princeton, NJ 08544
kulkarni@princeton.edu

**Robert E. Schapire**
Department of Computer Science
Princeton University
Princeton, NJ 08544
schapire@cs.princeton.edu

## Abstract

We study the statistical convergence and consistency of regularized Boosting methods, where the samples are not independent and identically distributed (i.i.d.) but come from empirical processes of stationary $\beta$-mixing sequences. Utilizing a technique that constructs a sequence of independent blocks close in distribution to the original samples, we prove the consistency of the composite classifiers resulting from a regularization achieved by restricting the 1-norm of the base classifiers' weights. When compared to the i.i.d. case, the nature of sampling manifests in the consistency result only through generalization of the original condition on the growth of the regularization parameter.

## 1   Introduction

A significant development in machine learning for classification has been the emergence of boosting algorithms [1]. Simply put, a boosting algorithm is an iterative procedure that combines weak prediction rules to produce a composite classifier, the idea being that one can obtain very precise prediction rules by combining rough ones. It was shown in [2] that AdaBoost, the most popular Boosting algorithm, can be seen as stage-wise fitting of additive models under the exponential loss function and it effectively minimizes an empirical loss function that differs from the probability of incorrect prediction. From this perspective, boosting can be seen as performing a greedy stage-wise minimization of various loss functions empirically. The question of whether boosting achieves Bayes-consistency then arises, since minimizing an empirical loss function does not necessarily imply minimizing the generalization error. When run a very long time, the AdaBoost algorithm, though resistant to overfitting, is not immune to it [2, 3]. There also exist cases where running Adaboost

forever leads to a prediction error larger than the Bayes error in the limit of infinite sample size. Consequently, one approach for the study of consistency is to modify the original Adaboost algorithm by imposing some constraints on the weights of the composite classifier to avoid overfitting. In this regularized version of Adaboost, the 1-norm of the weights of the base classifiers is restricted to a fixed value. The minimization of the loss function is performed over the restricted class [4, 5].

In this paper, we examine the convergence and consistency of regularized boosting algorithms with samples that are no longer i.i.d. but come from empirical processes of stationary weakly dependent sequences. A practical motivation for our study of non i.i.d. sampling is that in many learning applications observations are intrinsically temporal and hence often weakly dependent. Ignoring this dependency could seriously undermine the performance of the learning process (for instance, information related to the time-dependent ordering of samples would be lost). Recognition of this issue has led to several studies of non i.i.d. sampling [6, 7, 8, 9, 10, 11, 12].

To cope with weak dependence we apply mixing theory which, through its definition of mixing coefficients, offers a powerful approach to extend results for the traditional i.i.d. observations to the case of weakly dependent or mixing sequences. We consider the $\beta$-mixing coefficients, whose mathematical definition is deferred to Sec. 2.1. Intuitively, they provide a "measure" of how fast the dependence between the observations diminishes as the distance between them increases. If certain conditions on the mixing coefficients are satisfied to reflect a sufficiently fast decline in the dependence between observations as their distance grows, counterparts to results for i.i.d. random processes can be established. A comprehensive review of mixing theory results is provided in [13].

Our principal finding is that consistency of regularized Boosting methods can be established in the case of non-i.i.d. samples coming from empirical sequences of stationary $\beta$-mixing sequences. Among the conditions that guarantee consistency, the mixing nature of sampling appears only through a generalization of the one on the growth of the regularization parameter originally stated for the i.i.d. case [4].

## 2 Background and Setup

### 2.1 Mixing Sequences

Let $\underline{W} = (W_i)_{i \geq 1}$ be a strictly stationary sequence of random variables, each having the same distribution $P$ on $\mathcal{D} \subset R^d$. Let $\sigma_1^l = \sigma(W_1, W_2, \ldots, W_l)$ be the $\sigma$-field generated by $W_1, \ldots, W_l$. Similarly, let $\sigma_{l+k}^\infty = \sigma(W_{l+k}, W_{l+k+1}, \ldots, )$. The following mixing coefficients characterize how close to independent a sequence $\underline{W}$ is.

**Definition 1.** *For any sequence $\underline{W}$, the $\beta$-mixing[1] coefficient is defined by*
$$\beta_W(n) = \sup_k \mathbb{E} \sup \left\{ |P(A|\sigma_1^k) - P(A)| : A \in \sigma_{k+n}^\infty \right\},$$
*where the expectation is taken w.r.t. $\sigma_1^k$.*

Hence $\beta_W(n)$ quantifies the degree of dependence between 'future' observations and 'past' ones separated by a distance of at least $n$. In this study, we will assume that the sequences

we consider are algebraically $\beta$-mixing. This property implies that the dependence between observations decreases fast enough as the distance between them increases.

**Definition 2.** *A sequence $\underline{W}$ is called $\beta$-mixing if* $\lim_{n\to\infty}\beta_W(n) = 0$. *Further, it is algebraically $\beta$-mixing if there is a positive constant $r_\beta$ such that $\beta_W(n) = O\left(n^{-r_\beta}\right)$.*

The choice of $\beta$-mixing appears appropriate given previous results that showed "uniform convergence of empirical means uniformly in probability" and "probably approximately correct" properties to be preserved for $\beta$-mixing inputs [11]. Some examples of $\beta$-mixing sequences that fit naturally in a learning scenario are certain Markov processes and Hidden Markov Models [11]. In practice, if the mixing properties are unknown, they need to be estimated. Although it is difficult to find them in general, there exist simple methods to determine the mixing rates for various classes of random processes (e.g. Gaussian, Markov, ARMA, ARCH, GARCH). Hence the assumption of a known mixing rate is reasonable and has been adopted by many studies [6, 7, 8, 9, 10, 12].

### 2.2   Classification with Stationary $\beta$-Mixing Training Data

In the standard binary classification problem, the training data consist of a set $S_n = \{(X_1, Y_1),\ldots,(X_n, Y_n)\}$, where $X_k$ belongs to some measurable space $\mathcal{X}$, and $Y_k$ is in $\{-1, 1\}$. Using $S_n$, a classifier $h_n : \mathcal{X} \to \{-1, 1\}$ is built to predict the label $Y$ of an unlabeled observation $X$. Traditionally, the samples are assumed to be i.i.d., and to our knowledge, this assumption is made by all the studies on boosting consistency. In this paper, we suppose that the sampling is no longer i.i.d. but corresponds to an empirical process of stationary $\beta$-mixing sequences. More precisely, let $\mathcal{D} = \mathcal{X} \times \mathcal{Y}$, where $\mathcal{Y} = \{-1, +1\}$. Let $W_i = (X_i, Y_i)$. We suppose that $\underline{W} = (W_i)_{i\geq 1}$ is a strictly stationary sequence of random variables, each having the same distribution $P$ on $\mathcal{D}$ and that $\underline{W}$ is $\beta$-mixing (see Definition 2). This setup is in line with [7]. We assume that the unlabeled observation is such that $(X, Y)$ is independent of $S_n$ but with the same marginal.

## 3   Statistical Convergence and Consistency of Regularized Boosting for Stationary $\beta$-Mixing Sequences

### 3.1   Regularized Boosting

We adopt the framework of [4] which we now recall. Let $\mathcal{H}$ denote the class of base classifiers $h : \mathcal{X} \to \{-1, 1\}$, which usually consists of simple rules (for instance decision stumps). This class is required to have finite VC-dimension. Call $\mathcal{F}$, the class of functions $f : \mathcal{X} \to [-1, 1]$ obtained as convex combinations of the classifiers in $\mathcal{H}$:

$$\mathcal{F} = \left\{ f(X) = \sum_{j=1}^{t} \alpha_j h_j(X) : t \in \mathbb{N}, \alpha_1,\ldots,\alpha_t \geq 0, \sum_{j=1}^{t} \alpha_j = 1, h_1,\ldots,h_t \in \mathcal{H}\right\}.$$
(1)

Each $f_n \in \mathcal{F}$ defines a classifier $h_{f_n} = \mathrm{sign}(f_n)$ and for simplicity the generalization error $L(h_{f_n})$ is denoted by $L(f_n)$. Then the training error is denoted by $L_n(f_n) = 1/n \sum_{i=1}^{n} \mathbf{I}_{[h_{f_n}(X_i)\neq Y_i]}$. Define $Z(f) = -f(X)Y$ and $Z_i(f) = -f(X_i)Y_i$. Instead of minimizing the indicator of misclassification ($\mathbf{I}_{[-f(X)Y>0]}$), boosting methods are shown to effectively minimize a smooth convex cost function of $Z(f)$. For instance, Adaboost is based on the exponential function. Consider a positive, differentiable, strictly increasing, and strictly convex function $\phi : \mathbb{R} \to \mathbb{R}^+$ and assume that $\phi(0) = 1$ and that $\lim_{x\to-\infty}\phi(x) = 0$. The corresponding cost function and empirical cost function are respectively $C(f) = \mathbb{E}\phi(Z(f))$ and $C_n(f) = 1/n \sum_{i=1}^{n} \phi(Z_i(f))$. Note that $L(f) \leq C(f)$, since $\mathbb{I}_{[x>0]} \leq \phi(x)$.

The iterative aspect of boosting methods is ignored to consider only their performing an (approximate) minimization of the empirical cost function or, as we shall see, a series of cost functions. To avoid overfitting, the following regularization procedure is developed for the choice of the cost functions. Define $\phi_\lambda$ such that $\forall \lambda > 0$ $\phi_\lambda(x) = \phi(\lambda x)$. The corresponding empirical and expected cost functions become $C_n^\lambda(f) = \frac{1}{n} \sum_{i=1}^n \phi_\lambda(Z_i(f))$ and $C^\lambda(f) = \mathbb{E}\phi_\lambda(Z(f))$. The minimization of a series of cost functions $C^\lambda$ over the convex hull of $\mathcal{H}$ is then analyzed.

## 3.2 Statistical Convergence

The nature of the sampling intervenes in the following two lemmas that relate the empirical cost $C_n^\lambda(f)$ and true cost $C^\lambda(f)$.

**Lemma 1.** *Suppose that for any $n$, the training data $(X_1, Y_1), \ldots (X_n, Y_n)$ comes from a stationary algebraically $\beta$-mixing sequence with $\beta$-mixing coefficients $\beta(m)$ satisfying $\beta(m) = O(m^{-r_\beta})$, $m \in \mathbb{N}$ and $r_\beta$ a positive constant. Then for any $\lambda > 0$ and $b \in [0, 1)$,*

$$\mathbb{E} \sup_{f \in \mathcal{F}} |C^\lambda(f) - C_n^\lambda(f)| \leq 4\lambda\phi'(\lambda) \frac{c_1}{n^{(1-b)/2}} + 2\phi(\lambda)\left(\frac{1}{n^{b(1+r_\beta)-1}} + \frac{2}{n^{1-b}}\right). \quad (2)$$

**Lemma 2.** *Let the training data be as in Lemma 1. For any $b \in [0, 1)$, and $\alpha \in (0, 1-b)$, let $\epsilon_n = 3(2c_1 + n^{\alpha/2})\lambda\phi'(\lambda)/n^{(1-b)/2}$. Then for any $\lambda > 0$*

$$\mathbb{P}\left(\sup_{f \in \mathcal{F}} |C^\lambda(f) - C_n^\lambda(f)| > \epsilon_n\right) \leq \exp(-4c_2 n^\alpha) + O(n^{1-b(r_\beta+1)}). \quad (3)$$

The constants $c_1$ and $c_2$ in the above lemmas are given in the proofs of Lemma 1 (Section 4.2) and Lemma 2 (Section 4.3) respectively.

## 3.3 Consistency Result

The following summarizes the assumptions that are made to prove consistency.

**Assumption 1.**
*I- Properties of the sample sequence: The samples $(X_1, Y_1), \ldots, (X_n, Y_n)$ are assumed to come from a stationary algebraically $\beta$-mixing sequence with $\beta$-mixing coefficients $\beta_{X,Y}(n) = O(n^{-r_\beta})$, $r_\beta$ being a positive constant.*
*II- Properties of the cost function $\phi$: $\phi$ is assumed to be a differentiable, strictly convex, strictly increasing cost function such that $\phi(0) = 1$ and $\lim_{x \to -\infty} \phi(x) = 0$.*
*III- Properties of the base hypothesis space: $\mathcal{H}$ has finite VC dimension. The distribution of $(X, Y)$ and the class $\mathcal{H}$ are such that $\lim_{\lambda \to \infty} \inf_{f \in \lambda \mathcal{F}} C(f) = C^*$, where $\lambda\mathcal{F} = \{\lambda f : f \in \mathcal{F}\}$ and $C^* = \inf C(f)$ over all measurable functions $f : \mathcal{X} \to \mathbb{R}$.*
*IV- Properties of the smoothing parameter: We assume that $\lambda_1, \lambda_2, \ldots$ is a sequence of positive numbers satisfying $\lambda_n \to \infty$ as $n \to \infty$, and that there exists a constant $c \in \left(\frac{1}{1+r_\beta}, 1\right)$ such that $\lambda_n\phi'(\lambda_n)/n^{(1-c)/2} \to 0$ as $n \to \infty$.*

Call $\hat{f}_n^\lambda$ the function in $\mathcal{F}$ which approximatively minimizes $C_n^\lambda(f)$, i.e. $\hat{f}_n^\lambda$ is such that $C_n^\lambda(\hat{f}_n^\lambda) \leq \inf_{f \in \mathcal{F}} C_n^\lambda(f) + \epsilon_n = \inf_{f \in \mathcal{F}} \frac{1}{n} \sum_{i=1}^n \phi_\lambda(Z_i(f)) + \epsilon_n$, with $\epsilon_n \to 0$ as $n \to \infty$. The main result is the following.

**Theorem 1.** *Consistency of regularized boosting methods for stationary $\beta$-mixing sequences. Let $f_n = \hat{f}_n^{\lambda_n} \in \mathcal{F}$, where $\hat{f}_n^{\lambda_n}$ (approximatively) minimizes $C_n^{\lambda_n}(f)$. Under Assumption 1, $\lim_{n \to \infty} L(h_{f_n} = \text{sign}(f_n)) = L^*$ almost surely and $h_{f_n}$ is strongly Bayes-risk consistent.*

Cost functions satisfying Assumption 1.II include the exponential function and the logit function $\log_2(1 + e^x)$. Regarding Assumption 1.II, the reader is referred to [4](Remark on

(denseness assumption)). In Assumption 1.IV, notice that the nature of sampling leads to a generalization of the condition on the growth of $\lambda_n \phi'(\lambda_n)$ already present in the i.i.d. setting [4]. More precisely, the nature of sampling manifests through parameter $c$, which is limited by $r_\beta$. The assumption that $r_\beta$ is known is quite strict but cannot be avoided (for instance this assumption is widely made in the field of time series analysis). On a positive note, if unknown, $r_\beta$ can be determined for various classes of processes as mentioned Section 2.1.

## 4 Proofs

### 4.1 Preparation to the Proofs: the Blocking Technique

The key issue resides in upper bounding

$$\sup_{f \in \mathcal{F}} \left| C_n^\lambda(f) - C^\lambda(f) \right| = \sup_{f \in \mathcal{F}} \left| 1/n \sum_{i=1}^n \phi\left(-\lambda f(X_i) Y_i\right) - \mathbb{E}\phi\left(-\lambda f(X_1) Y_1\right) \right|, \quad (4)$$

where $\mathcal{F}$ is given by (1). Let $W = (X, Y)$, $W_i = (X_i, Y_i)$. Define the function $g_\lambda$ by $g_\lambda(W) = g_\lambda(X, Y) = \phi\left(-\lambda f(X) Y\right)$ and the class $\mathcal{G}_\lambda$ by $\mathcal{G}_\lambda = \{g_\lambda : g_\lambda(X, Y) = \phi\left(-\lambda f(X) Y\right), f \in \mathcal{F}\}$. Then (4) can be rewritten as

$$\sup_{f \in \mathcal{F}} \left| C_n^\lambda(f) - C^\lambda(f) \right| = \sup_{g_\lambda \in \mathcal{G}_\lambda} \left| n^{-1} \sum_{i=1}^n g_\lambda(W_i) - \mathbb{E}g_\lambda(W_1) \right|.$$

Note that the class $\mathcal{G}_\lambda$ is uniformly bounded by $\phi(\lambda)$. Besides, if $\mathcal{H}$ is a class of measurable functions, then $\mathcal{G}_\lambda$ is also a class of measurable functions, by measurability of $\mathcal{F}$.

As the $W_i$'s are not i.i.d, we propose to use the blocking technique developed in [12, 14] to construct i.i.d blocks of observations which are close in distribution to the original sequence $W_1, \ldots, W_n$. This enables us to work on the sequence of independent blocks instead of the original sequence. We use the same notation as in [12]. The protocol is the following. Let $(b_n, \mu_n)$ be a pair of integers, such that

$$(n - 2b_n) \le 2b_n \mu_n \le n. \quad (5)$$

Divide the segment $W_1 = (X_1, Y_1), \ldots, W_n = (X_n, Y_n)$ of the mixing sequence into $2\mu_n$ blocks of size $b_n$, followed by a remaining block (of size at most $2b_n$). Consider the odd blocks only. If their size $b_n$ is large enough, the dependence between them is weak, since two odd blocks are separated by an even block of the same size $b_n$. Therefore, the odd blocks can be approximated by a sequence of independent blocks with the same within-block structure. The same holds if we consider the even blocks. Let $(\xi_1, \ldots, \xi_{b_n}), (\xi_{b_n+1}, \ldots, \xi_{2b_n}), \ldots, \left(\xi_{(2\mu_n-1)b_n}, \ldots, \xi_{2\mu_n b_n}\right)$ be independent blocks such that $\left(\xi_{jb_n+1}, \ldots, \xi_{(j+1)b_n}\right) =_{\mathcal{D}} \left(W_{jb_n+1}, \ldots, W_{(j+1)b_n}\right)$, for $j = 0, \ldots, \mu_n - 1$.
For $j = 1, \ldots, 2\mu_n$, and any $g \in \mathcal{G}_\lambda$, define
$Z_{j,g} := \sum_{i=(j-1)b_n+1}^{jb_n} g(\xi_i) - b_n \mathbb{E}g(\xi_1), \quad \tilde{Z}_{j,g} := \sum_{i=(j-1)b_n+1}^{jb_n} g(W_i) - b_n \mathbb{E}g(W_1)$.
Let $\mathcal{O}_{\mu_n} = \{1, 3, \ldots, 2\mu_n - 1\}$ and $\mathcal{E}_{\mu_n} = \{2, 4, \ldots, 2\mu_n\}$.
Define $Z_{i,j}(f)$ as $Z_{i,j}(f) := -f\left(\xi_{(2j-2)b_n+i,1}\right) \cdot \xi_{(2j-2)b_n+i,2}$, where $\xi_{k,1}$ and $\xi_{k,2}$ are respectively the 1st and 2nd coordinate of the vector $\xi_k$. These correspond to the $Z_k(f) = -f(X_k) Y_k$ for $k$ in the odd blocks $1, \ldots, b_n, 2b_n + 1, \ldots, 3bn, \ldots$.

### 4.2 Proof sketch of Lemma 1

**A. Working with Independent Blocks.** We show that

$$\mathbb{E} \sup_{g \in \mathcal{G}_\lambda} \left| \frac{1}{n} \sum_{i=1}^n g(W_i) - \mathbb{E}g(W_1) \right| \le 2\mathbb{E} \sup_{g \in \mathcal{G}_\lambda} \left| \frac{1}{n} \sum_{j \in \mathcal{O}_{\mu_n}} Z_{j,g} \right| + \phi(\lambda)\left(\mu_n \beta_W(b_n) + \frac{2b_n}{n}\right). \quad (6)$$

**Proof.** Without loss of generality, assume that $\mathbb{E}g\left(W_1\right) = \mathbb{E}g\left(\xi_1\right) = 0$.
Then, $\mathbb{E}\sup_g \left|\frac{1}{n}\sum_{i=1}^n g\left(W_i\right)\right| = \mathbb{E}\sup_g \left|\frac{1}{n}\left(\sum_{\mathcal{O}_{\mu_n}} \tilde{Z}_{j,g} + \sum_{\mathcal{E}_{\mu_n}} \tilde{Z}_{j,g} + R\right)\right|$, where $R$
is the remainder term consisting of a sum of at most $2b_n$ terms. Noting that $\forall g \in$
$\mathcal{G}_\lambda$, $|g| \leq \phi\left(\lambda\right)$, it follows that $\mathbb{E}\sup_g |\frac{1}{n}\sum_{i=1}^n g\left(W_i\right)| \leq \mathbb{E}(\sup_g |\frac{1}{n}\sum_{\mathcal{O}_{\mu_n}} \tilde{Z}_{j,g}|) +$
$\mathbb{E}(\sup_g |\frac{1}{n}\sum_{\mathcal{E}_{\mu_n}} \tilde{Z}_{j,g}|) + \frac{\phi(\lambda)(2b_n)}{n}$. We use the following intermediary lemma.

**Lemma 3 (adapted from [15], Lemma 4.1).** *Call* $\mathbf{Q}$ *the distribution of* $(W_1,\ldots,W_{b_n},W_{2b_n+1},\ldots,W_{3b_n},\ldots)$ *and* $\widetilde{\mathbf{Q}}$ *the distribution of* $(\xi_1,\ldots,\xi_{b_n},\xi_{2b_n+1},\ldots,\xi_{3b_n},\ldots)$. *For any measurable function* $h$ *on* $\mathbb{R}^{b_n\mu_n}$ *with bound* $H$, $\left|\mathbf{Q}h\left(W_1,\ldots\right) - \widetilde{\mathbf{Q}}h\left(\xi_1,\ldots\right)\right| \leq H\left(\mu_n - 1\right)\beta_W\left(b_n\right)$. *The same result holds for* $(W_{b_n+1},\ldots,W_{2b_n},W_{3b_n+1},\ldots,W_{4b_n}\ldots)$.

Using this with $h(W_1,\ldots) = \sup_g |\frac{1}{n}\sum_{\mathcal{O}_{\mu_n}} \tilde{Z}_{j,g}|$ and $h(W_{b_n+1},\ldots) = \sup_g |\frac{1}{n}\sum_{\mathcal{E}_{\mu_n}} \tilde{Z}_{j,g}|$
respectively, and noting that $H = \phi\left(\lambda\right)/2$, we have $\mathbb{E}\sup_g |\frac{1}{n}\sum_{i=1}^n g\left(W_i\right)| \leq$
$\mathbb{E}\sup_g |\frac{1}{n}\sum_{\mathcal{O}_{\mu_n}} Z_{j,g}| + \frac{\phi(\lambda)}{2}\mu_n\beta_W\left(b_n\right) + \mathbb{E}\sup_g |\frac{1}{n}\sum_{\mathcal{E}_{\mu_n}} Z_{j,g}| + \frac{\phi(\lambda)}{2}\mu_n\beta_W\left(b_n\right) + \frac{\phi(\lambda)(2b_n)}{n}$.
As the $Z_{j,g}$'s from odd and even blocks have the same distribution, we obtain (6). $\square$

**B. Symmetrization.** The odd blocks $Z_{j,g}$'s being independent, we can use the standard
symmetrization techniques. Let $Z'_{j,g}$'s be i.i.d. copies of the $Z_{j,g}$'s. Let $Z'_{i,j}(f)$'s be the
corresponding copies of the $Z_{i,j}(f)$. Let $(\sigma_i)$ be a Rademacher sequence, i.e. a sequence
of independent random variables taking the values $\pm 1$ with probability $1/2$. Then by [16],
Lemma 6.3 (Proof is omitted due to space constraints), we have

$$\mathbb{E}\sup_g \left|\frac{1}{n}\sum_{j\in\mathcal{O}_{\mu_m}} Z_{j,g}\right| \leq \mathbb{E}\sup_g \left|\frac{1}{n}\sum_{j\in\mathcal{O}_{\mu_n}} \sigma_j\left(Z_{j,g} - Z'_{j,g}\right)\right|. \tag{7}$$

**C. Contraction Principle.** We now show that

$$\mathbb{E}\sup_{g\in\mathcal{G}_\lambda} \left|\frac{1}{n}\sum_{j\in\mathcal{O}_{\mu_n}} Z_{j,g}\right| \leq 2\cdot b_n\lambda\phi'\left(\lambda\right)\mathbb{E}\sup_{f\in\mathcal{F}} \left|\frac{1}{n}\sum_{j=1}^{\mu_n}\sigma_j Z_{1,j}(f)\right|. \tag{8}$$

**Proof.** As $Z_{j,g} = \sum_{i=1}^{b_n}\phi_\lambda(Z_{i,j}(f))$, and the $Z_{i,j}(f)$'s and $Z'_{i,j}(f)$'s are i.i.d., with (7)
$\mathbb{E}\sup_g |\frac{1}{n}\sum_{j\in\mathcal{O}_{\mu_n}} Z_{j,g}| \leq \mathbb{E}\sup_g |\frac{1}{n}\sum_{j=1}^{\mu_n}\sigma_j\sum_{i=1}^{b_n}\left(\phi_\lambda\left(Z_{i,j}(f)\right) - \phi_\lambda\left(Z'_{i,j}(f)\right)\right)| \leq$
$2b_n\mathbb{E}\sup_g |\frac{1}{n}\sum_{j=1}^{\mu_n}\sigma_j\left(\phi_\lambda\left(Z_{1,j}(f)\right) - 1\right)|$. By applying the "Comparison Theorem", Theorem 7 in [17], to the contraction $\psi\left(x\right) = \left(1/\lambda\phi'\left(\lambda\right)\right)\left(\phi_\lambda\left(x\right) - 1\right)$, we obtain (8). $\square$

**D. Maximal Inequality.** We show that there exists a constant $c_1 > 0$ such that

$$\mathbb{E}\sup_{f\in\mathcal{F}} \left|\frac{1}{n}\sum_{j=1}^{\mu_n}\sigma_j Z_{1,j}(f)\right| \leq \frac{c_1\sqrt{\mu_n}}{n}. \tag{9}$$

**Proof.** Denote $(h_1,\ldots,h_N)$ by $h_1^N$. One can write $\mathbb{E}\sup_{f\in\mathcal{F}} |\frac{1}{n}\sum_{j=1}^{\mu_n}\sigma_j Z_{1,j}(f)| =$
$\frac{1}{n}\mathbb{E}\sup_{N\geq 1}\sup_{h_1^N\in\mathcal{H}^N}\sup_{\alpha_1,\ldots,\alpha_N} |\sum_{j=1}^{\mu_n}\sum_{k=1}^N \alpha_k\sigma_j\xi_{(1,j),2}h_k\left(\xi_{(2j-2)b_n+1,1}\right)|$. Since
$\xi_{(2j-2)b_n+1,2}$ and $\xi_{(2j'-2)b_n+1,2}$ are i.i.d. for all $j \neq j'$ (they come from different blocks),
and $(\sigma_j)$ is a Rademacher sequence, then $\left(\sigma_j\xi_{(2j-2)b_n+1,2}h_k\left(\xi_{(2j-2)b_n+1,1}\right)\right)_{j=1,\ldots,\mu_n}$
has the same distribution as $\left(\sigma_j h_k\left(\xi_{(2j-2)b_n+1,1}\right)\right)_{j=1,\ldots,\mu_n}$. Hence

$$\mathbb{E}\sup_{f\in\mathcal{F}} \left|\frac{1}{n}\sum_{j=1}^{\mu_n}\sigma_j Z_{1,j}(f)\right| = \frac{1}{n}\mathbb{E}\sup_{N\geq 1}\sup_{h_1^N\in\mathcal{H}^N}\sup_{\alpha_1,\ldots,\alpha_N} \left|\sum_{j=1}^{\mu_n}\sum_{k=1}^N \sigma_j\alpha_k h_k\left(\xi_{(2j-2)b_n+1,1}\right)\right|.$$

By the same argument as used in [4], p.53 on the maximum of a linear function over
a convex polygon, the supremum is achieved when $\alpha_k = 1$ for some $k$. Hence we get

$\mathbb{E}\sup_{f\in\mathcal{F}}\left|\frac{1}{n}\sum_{j=1}^{\mu_n}\sigma_j Z_{1,j}(f)\right| = \frac{1}{n}\mathbb{E}\sup_{h\in\mathcal{H}}\left|\sum_{j=1}^{\mu_n}\sigma_j h\left(\xi_{(1,j),1}\right)\right|$. Noting that for all $j \neq j'$, $h(\xi_{(2j-2)b_n+1,1})$ and $h(\xi_{(2j'-2)b_n+1,1})$ are i.i.d. and that Rademacher processes are sub-gaussian, we have by [18], Corollary 2.2.8

$$\frac{1}{n}\mathbb{E}\sup_{h\in\mathcal{H}}\left|\sum_{j=1}^{\mu_n}\sigma_j h\left(\xi_{(2j-2)b_n+1,1}\right)\right| \leq \frac{1}{n}\mathbb{E}\sup_{h\in\mathcal{H}\cup\{0\}}\left|\sum_{j=1}^{\mu_n}\sigma_j h\left(\xi_{(2j-2)b_n+1,1}\right)\right|$$

$$\leq \frac{c'\sqrt{\mu_n}}{n}\int_0^\infty (\log\sup_P N\left(\epsilon,\rho_{2,P_n},\mathcal{H}\cup\{0\}\right))^{1/2}d\epsilon,$$

where $c'$ is a constant and $N\left(\epsilon,\rho_{2,P_n},\mathcal{H}\cup\{0\}\right)$ is the empirical $\mathcal{L}_2$ covering number. As $\mathcal{H}$ has finite VC-dimension (see Assumption 1.III), there exists a positive constant $w$ such that $\sup_P N(\epsilon,\rho_{2,P_n},\mathcal{H}\cup\{0\}) = O_P(\epsilon^{-w})$(see [18], Theorem 2.6.1). Hence $\int_0^\infty (\log\sup_{P_n} N\left(\epsilon,\rho_{2,P_n},\mathcal{H}\cup\{0\}\right))^{1/2}d\epsilon < \infty$. and (9) follows. $\square$

**E. Establishing (2).** Combining (6),(8), and (9), we have
$\mathbb{E}\sup_{g\in\mathcal{G}_\lambda}\left|\frac{1}{n}\sum_{i=1}^n g\left(W_i\right) - \mathbb{E}g\left(W_1\right)\right| \leq 4b_n\lambda\phi'\left(\lambda\right)\frac{c_1\sqrt{\mu_n}}{n} + \phi\left(\lambda\right)\left(\mu_n\beta_W\left(b_n\right) + \frac{2b_n}{n}\right)$.
Take $b_n = n^b$, with $0 \leq b < 1$. By (5), we obtain $\mu_n \leq n^{1-b}/2$. Besides, as we assumed that the sequence $\underline{W}$ is algebraically $\beta$-mixing (see Definition 2), $\beta_W\left(n\right) = O\left(n^{-r_\beta}\right)$. Then $\mu_n\beta_W\left(b_n\right) = O\left(n^{1-b(1+r_\beta)}\right)$, and we arrive at (2). $\blacksquare$

### 4.3 Proof Sketch of Lemma 2
**A. Working with Independent Blocks and Symmetrization.** For any $b \in [0,1), \alpha \in (0, 1-b)$, let
$$\epsilon_n = 3(2c_1 + n^{\alpha/2})\lambda\phi'(\lambda)/n^{(1-b)/2}. \tag{10}$$
We show
$$\mathbb{P}\left(\sup_{g\in\mathcal{G}_\lambda}\left|\frac{1}{n}\sum_{i=1}^n g\left(W_i\right) - \mathbb{E}g\left(W_1\right)\right| > \epsilon_n\right) \leq 2\mathbb{P}\left(\sup_{g\in\mathcal{G}_\lambda}\left|\frac{1}{n}\sum_{j\in\mathcal{O}_{\mu_n}} Z_{j,g}\right| > \epsilon_n/3\right) + O(n^{1-b(1+r_\beta)}). \tag{11}$$

**Proof.** By [12], Lemma 3.1, we have that for any $\epsilon_n$ such that $\phi(\lambda)b_n = o(n\epsilon_n)$, $\mathbb{P}\left(\sup_{g\in\mathcal{G}_\lambda}\left|\frac{1}{n}\sum_{i=1}^n g\left(W_i\right) - \mathbb{E}g\left(W_1\right)\right| > \epsilon_n\right) \leq 2\mathbb{P}\left(\sup_{g\in\mathcal{G}_\lambda}\left|\frac{1}{n}\sum_{j\in\mathcal{O}_{\mu_n}} Z_{j,g}\right| > \epsilon_n/3\right) + 4\mu_n\beta_W(b_n)$. Set $b_n = n^b$, with $0 \leq b < 1$. Then $\mu_n\beta_W(b_n) = O(n^{1-b(1+r_\beta)})$ (for the same reasons as in Section 4.2 E.). With $\epsilon_n$ as in (10), and since Assumption 1.II implies that $\lambda\phi'(\lambda) \geq \phi(\lambda) - 1$, we automatically obtain $\phi(\lambda)b_n = o(n\epsilon_n)$. $\square$

**B. McDiarmid's Bounded Difference Inequality.** For $\epsilon_n$ as in (10), there exists a constant $c_2 > 0$ such that,
$$\mathbb{P}\left(\sup_{g\in\mathcal{G}_\lambda}\left|\frac{1}{n}\sum_{j\in\mathcal{O}_{\mu_n}} Z_{j,g}\right| > \epsilon_n/3\right) \leq \exp(-4c_2 n^\alpha). \tag{12}$$

**Proof.** The $Z_{j,g}$'s of the odd block being independent, we can apply McDiarmid's bounded difference inequality ([19], Theorem 9.2 p.136) on the function $\sup_{g\in\mathcal{G}_\lambda}\left|\frac{1}{n}\sum_{j\in\mathcal{O}_{\mu_n}} Z_{j,g}\right|$ which depends of $Z_{1,g}, Z_{3,g}\ldots, Z_{2\mu_n-1,g}$. Noting that changing the value of one variable does not change the value of the function by more that $b_n\phi\left(\lambda\right)/n$, we obtain with $b_n = n^b$ that for all $\epsilon > 0$,
$\mathbb{P}\left(\sup_{g\in\mathcal{G}_\lambda}\left|\frac{1}{n}\sum_{j\in\mathcal{O}_{\mu_n}} Z_{j,g}\right| > \mathbb{E}\sup_{g\in\mathcal{G}_\lambda}\left|\frac{1}{n}\sum_{j\in\mathcal{O}_{\mu_n}} Z_{j,g}\right| + \epsilon\right) \leq \exp\left(\frac{-4\epsilon^2 n^{1-b}}{\phi(\lambda)^2}\right)$.
Combining (8) and (9) from the proof of Lemma 1, and with $b_n = n^b$, we have $\mathbb{E}\sup_{g\in\mathcal{G}_\lambda}\left|\frac{1}{n}\sum_{j\in\mathcal{O}_{\mu_n}} Z_{j,g}\right| \leq 2\lambda\phi'\left(\lambda\right)C/n^{(1-b)/2}$. With $\epsilon = n^{\alpha/2}\lambda\phi'(\lambda)/n^{(1-b)/2}$, we obtain $\epsilon_n$ as in (10). Pick $\lambda_0$ such that $0 < \lambda_0 < \lambda$. Then, since $\lambda\phi'(\lambda) \geq \phi(\lambda) - 1$, (12) follows with $c_2 = (1 - 1/\phi(\lambda_0))^2$. $\square$

**C. Establishing (3).** Combining (11) and (12) we obtain (3). ∎

### 4.4   Proof Sketch of Theorem 1

Let $\bar{f}_\lambda$ a function in $\mathcal{F}$ minimizing $C^\lambda$. With $f_n = \hat{f}_n^{\lambda_n}$, we have

$$C\left(\lambda_n f_n\right) - C^* = \left(C^{\lambda_n}(\hat{f}_n^{\lambda_n}) - C^{\lambda_n}(\bar{f}_{\lambda_n})\right) + \left(\inf_{f \in \lambda_n \mathcal{F}} C(f) - C^*\right).$$

Since $\lambda_n \to \infty$, the second term on the right-hand side converges to zero by Assumption 1.III. By [19], Lemma 8.2, we have $C^{\lambda_n}(\hat{f}_n^{\lambda_n}) - C^{\lambda_n}\left(\bar{f}_{\lambda_n}\right) \le 2\sup_{f \in \mathcal{F}} |C^{\lambda_n}(f) - C_n^{\lambda_n}(f)|$. By Lemma 2, $\sup_{f \in \mathcal{F}} |C^{\lambda_n}(f) - C_n^{\lambda_n}(f)| \to 0$ with probability 1 if, as $n \to \infty$, $\lambda_n \phi'(\lambda_n) n^{(\alpha+b-1)/2} \to 0$ and $b > 1/(1 + r_\beta)$. Hence if Assumption 1.IV holds, $C\left(\lambda_n f_n\right) \to C^*$ with probability 1. By [4], Lemma 5, the theorem follows. ∎

## Footnotes

[1]To gain insight into the notion of $\beta$-mixing, it is useful to think of the $\sigma$-field generated by a random variable $X$ as the "body of information" carried by $X$. This leads to the following interpretation of $\beta$-mixing. Suppose that the index $i$ in $W_i$ is the time index. Let $A$ be an event happening in the future within the period of time between $t = k + n$ and $t = \infty$. $|P(A|\sigma_1^k) - P(A)|$ is the absolute difference between the probability that event $A$ occurs, given the knowledge of the information generated by the past up to $t = k$, and the probability of event $A$ occurring without this knowledge. Then, the greater the dependence between $\sigma_1^k$ (the information generated by $(W_1, \ldots, W_k)$) and $\sigma_{k+n}^\infty$ (the information generated by $(W_{k+n}, \ldots, W_\infty)$), the larger the coefficient $\beta_W(n)$.

## References

[1] Schapire, R.E.: The Boosting Approach to Machine Learning An Overview. In Proc. of the MSRI Workshop on Nonlinear Estimation and Classification (2002)

[2] Friedman, J., Hastie T., Tibshirani, R.: Additive logistic regression: A statistical view of boosting. Ann. Statist. **38** (2000) 337–374

[3] Jiang, W.: Does Boosting Overfit:Views From an Exact Solution. Technical Report **00-03** Department of Statistics, Northwestern University (2000)

[4] Lugosi, G., Vayatis, N.: On the Bayes-risk consistency of boosting methods. Ann. Statist. **32** (2004) 30–55

[5] Zhang, T.: Statistical Behavior and Consistency of Classification Methods based on Convex Risk Minimization. Ann. Statist. **32** (2004) 56–85

[6] Györfi, L., Härdle, W., Sarda, P., and Vieu, P.: Nonparametric Curve Estimation from Time Series. Lecture Notes in Statistics. Springer-Verlag, Berlin. (1989)

[7] Irle, A.: On the consistency in nonparametric estimation under mixing assumptions. J. Multivariate Anal. **60** (1997) 123–147

[8] Meir, R.: Nonparametric Time Series Prediction Through Adaptative Model Selection. Machine Learning **39** (2000) 5–34

[9] Modha, D., Masry, E.: Memory-Universal Prediction of Stationary Random Processes. IEEE Trans. Inform. Theory **44** (1998) 117–133

[10] Roussas, G.G.: Nonparametric estimation in mixing sequences of random variables. J. Statist. Plan. Inference. **18** (1988) 135–149

[11] Vidyasagar, M.: A Theory of Learning and Generalization: With Applications to Neural Networks and Control Systems. Second Edition. Springer-Verlag, London (2002)

[12] Yu, B.: Density estimation in the $L^\infty$ norm for dependent data with applications. Ann. Statist. **21** (1993) 711–735

[13] Doukhan, P.: Mixing Properties and Examples. Springer-Verlag, New York (1995)

[14] Yu, B.: Some Results on Empirical Processes and Stochastic Complexity. Ph.D. Thesis, Dept of Statistics, U.C. Berkeley (Apr. 1990)

[15] Yu, B.: Rate of convergence for empirical processes of stationary mixing sequences. Ann. Probab. **22** (1994) 94–116.

[16] Ledoux, M., Talagrand, N.: Probability in Banach Spaces. Springer, New York (1991)

[17] Meir, R., Zhang, T.:Generalization error bounds for Bayesian mixture algorithms. J. Machine Learning Research (2003)

[18] van der Vaart, A.W., Wellner, J.A.: Weak convergence and empirical processes. Springer Series in Statistics. Springer-Verlag, New York (1996)

[19] Devroye, L., Györfi L., Lugosi, G.: A Probabilistic Theory of Pattern Recognition. Springer, New York (1996)
